# Neural System Model of Human Sound Localization

**Craig T. Jin**
Department of Physiology and
Department of Electrical Engineering,
Univ. of Sydney, NSW 2006, Australia

**Simon Carlile**
Department of Physiology
and Institute of Biomedical Research,
Univ. of Sydney, NSW 2006, Australia

## Abstract

This paper examines the role of biological constraints in the human auditory localization process. A psychophysical and neural system modeling approach was undertaken in which performance comparisons between competing models and a human subject explore the relevant biologically plausible "realism constraints". The directional acoustical cues, upon which sound localization is based, were derived from the human subject's head-related transfer functions (HRTFs). Sound stimuli were generated by convolving bandpass noise with the HRTFs and were presented to both the subject and the model. The input stimuli to the model was processed using the Auditory Image Model of cochlear processing. The cochlear data was then analyzed by a time-delay neural network which integrated temporal and spectral information to determine the spatial location of the sound source. The combined cochlear model and neural network provided a system model of the sound localization process. Human-like localization performance was qualitatively achieved for broadband and bandpass stimuli when the model architecture incorporated frequency division (or tonotopicity), and was trained using variable bandwidth and center-frequency sounds.

## 1 Introduction

The ability to accurately estimate the location of a sound source has obvious evolutionary advantages in terms of avoiding predators and finding prey. Indeed, humans are very accurate in their ability to localize broadband sounds. There has been a considerable amount of psychoacoustical research into the auditory processes involved in human sound localization (recent review [1]). Furthermore, numerous models of the human and animal sound localization process have been proposed (recent reviews [2, 3]). However, there still remains a large gap between the psychophysical and the model explanations. Principal congruence between the two approaches exists for localization performance under restricted conditions, such as for narrowband sounds where spectral integration is not required, or for restricted regions of space. Unfortunately, there is no existing computational model that accounts well for human sound localization performance for a wide-range of sounds (e.g., varying in bandwidth and center-frequency). Furthermore, the biological constraints pertinent to sound localization have generally not been explored by these models. These include the spectral resolution of the auditory system in terms of the number and bandwidth of

frequency channels and the role of tonotopic processing. In addition, the performance requirements of such a system are substantial and involve, for example, the accomodation of spectrally complex sounds, the robustness to irregularity in the sound source spectrum, and the channel based structure of spatial coding as evidenced by auditory spatial after-effects [4]. The crux of the matter is the notion that "biologically-likely realism", if built into a model, provides for a better understanding of the underlying processes.

This work attempts to bridge part of this gap between the modeling and psychophysics. It describes the development and use (for the first time, to the authors' knowledge) of a time-delay neural network model that integrates both spectral and temporal cues for auditory sound localization and compares the performance of such a model with the corresponding human psychophysical evidence.

## 2   Sound Localization

The sound localization performance of a normal hearing human subject was tested using stimuli consisting of three different band-passed sounds: (1) a low-passed sound (300 – 2000 Hz) (2) a high-passed sound (2000 – 14000 Hz) and (3) a broadband sound (300 – 14000 Hz). These frequency bands respectively cover conditions in which either temporal cues, spectral cues, or both dominate the localization process (see [1]). The subject performed five localization trials for each sound condition, each with 76 test locations evenly distributed about the subject's head. The detailed methods used in free-field sound localization can be found in [5]. A short summary is presented below.

### 2.1   Sound Localization Task

Human sound localization experiments were carried out in a darkened anechoic chamber. Free-field sound stimuli were presented from a loudspeaker carried on a semicircular robotic arm. These stimuli consisted of "fresh" white Gaussian noise appropriately band-passed for each trial. The robotic arm allowed for placement of the speaker at almost any location on the surface of an imaginary sphere, one meter in radius, centered on the subject's head. The subject indicated the location of the sound source by pointing his nose in the perceived direction of the sound. The subject's head orientation was monitored using an electromagnetic sensor system (Polhemus, Inc.).

### 2.2   Measurement and Validation of Outer Ear Acoustical Filtering

The cues for sound localization depend not only upon the spectral and temporal properties of the sound stimulus, but also on the acoustical properties of the individual's outer ears. It is generally accepted that the relevant acoustical cues (i.e., the interaural time difference, ITD; interaural level difference, ILD; and spectral cues) to a sound's location in the free-field are described by the head-related transfer function (HRTF) which is typically represented by a finite-length impulse response (FIR) filter [1]. Sounds filtered with the HRTF should be localizable when played over ear-phones which bypass the acoustical filtering of the outer ear. The illusion of free-field sounds using head-phones is known as virtual auditory space (VAS).

Thus in order to incorporate outer ear filtering into the modelling process, measurements of the subject's HRTFs were carried out in the anechoic chamber. The measurements were made for both ears simultaneously using a "blocked ear" technique [1]. 393 measurements were made at locations evenly distributed on the sphere. In order to establish that the HRTFs appropriately indicated the direction of a sound source the subject repeated the localization task as above with the stimulus presented in VAS.

### 2.3 Human Sound Localization Performance

The sound localization performance of the human subject in three different stimulus conditions (broadband, high-pass, low-pass) was examined in both the free-field and in virtual auditory space. Comparisons between the two (using correlational statistics, data not shown, but see [3]) across all sound conditions demonstrated their equivalence. Thus the measured HRTFs were highly effective.

Localization data across all three sound conditions (single trial VAS data shown in Fig. 1a) shows that the subject performed well in both the broadband and high-pass sound conditions and rather poorly in the low-pass condition, which is consistent with other studies [6]. The data is illustrated using spherical localization plots which well demonstrates the global distribution of localization responses. Given the large qualitative differences in the data sets presented below, this visual method of analysis was sufficient for evaluating the competing models. For each condition, the target and response locations are shown for both the left (L) and right (R) hemispheres of space. It is clear that in the low-pass condition, the subject demonstrated gross mislocalizations with the responses clustering toward the lower and frontal hemispheres. The gross mislocalizations correspond mainly to the traditional cone of confusion errors [6].

## 3 Localization Model

The sound localization model consisted of two basic system components: (1) a modified version of the physiological Auditory Image Model [7] which simulates the spectro-temporal characteristics of peripheral auditory processing, and (2) the computational architecture of a time-delay neural network. The sounds presented to the model were filtered using the subject's HRTFs in exactly the same manner as was used in producing VAS. Therefore, the modeling results can be compared with human localization performance on an individual basis.

The modeling process can be broken down into four stages. In the first stage a sound stimulus was generated with specific band-pass characteristics. The sound stimulus was then filtered with the subject's right and left ear HRTFs to render an auditory stimulus originating from a particular location in space. The auditory stimulus was then processed by the Auditory Image Model (AIM) to generate a neural activity profile that simulates the output of the inner hair cells in the organ of Corti and indicates the spiking probability of auditory nerve fibers. Finally, in the fourth and last stage, a time-delay neural network (TDNN) computed the spatial direction of the sound input based on the distribution of neural activity calculated by AIM.

A detailed presentation of the modeling process can be found in [3], although a brief summary is presented here. The distribution of cochlear filters across frequency in AIM was chosen such that the minimum center frequency was 300 Hz and the maximum center frequency was 14 kHz with 31 filters essentially equally spaced on a logarithmic scale. In order to fully describe a computational layer of the TDNN, four characteristic numbers must be specified: (1) the number of neurons; (2) the kernel length, a number which determines the size of the current layer's time-window in terms of the number of time-steps of the previous layer; (3) the kernel width, a number which specifies how many neurons in the previous layer with which there are actual connections; and (4) the undersampling factor, a number describing the multiplicative factor by which the current layer's time-step interval is increased from the previous layer's. Using this nomenclature, the architecture of the different layers of one TDNN is summarized in Table 1, with the smallest time-step being 0.15 ms. The exact connection arrangement of the network is described in the next section.

Table 1: The Architecture of the TDNN.

| Layer | Neurons | Kernel Length | Kernel Width | Undersampling |
|---|---|---|---|---|
| Input | 62 | — | — | — |
| Hidden 1 | 50 | 15 | 6 | 2 |
| Hidden 2 | 28 | 10 | 4, 5, 6 | 2 |
| Output | 393 | 4 | 28 | 1 |

The spatial location of a sound source was encoded by the network as a distributed response with the peak occurring at the output neuron representing the target location of the input sound. The output response would then decay away in the form of a two-dimensional Gaussian as one moves to neurons further away from the target location. This derives from the well-established paradigm that the nervous system uses overlapping receptive fields to encode properties of the physical world.

### 3.1 Networks with Frequency Division and Tonotopicity

The major auditory brainstem nuclei demonstrate substantial frequency division within their structure. The tonotopic organization of the primary auditory nerve fibers that innervate the cochlea carries forward to the brainstem's auditory nuclei. This arrangement is described as a tonotopic organization. Despite this fact and to our knowledge, no previous network model for sound localization incorporates such frequency division within its architecture. Typically (e.g., [8]) all of the neurons in the first computational layer are fully connected to *all* of the input cochlear frequency channels. In this work, different architectures were examined with varying amounts of frequency division imposed upon the network structure. The network with the architecture described above had its network connections constrained by frequency in a tonotopic like arrangement. The 31 input cochlear frequency channels for each ear were split into ten overlapping groups consisting generally of six contiguous frequency channels. There were five neurons in the first hidden layer for each group of input channels. The kernel widths of these neurons were set, not to the total number of frequency channels in the input layer, but only to the six contiguous frequency channels defining the group. Information across the different groups of frequency channels was progressively integrated in the higher layers of the network.

### 3.2 Network Training

Sounds with different center-frequency and bandwidth were used for training the networks. In one particular training paradigm, the center-frequency and bandwidth of the noise were chosen randomly. The center-frequency was chosen using a uniform probability distribution on a logarithmic scale that was similar to the physiological distribution of output frequency channels from AIM. In this manner, each frequency region was trained equally based on the density of neurons in that frequency region. During training, the error back-propagation algorithm was used with a summed squared error measure. It is a natural feature of the learning rule that a given neuron's weights are only updated when there is activity in its respective cochlear channels. So, for example, a training sound containing only low frequencies will not train the high-frequency neurons and vice versa. All modeling results correspond with a *single* tonotopically organized TDNN trained using random sounds (unless explicitly stated otherwise).

## 4 Localization Performance of a Tonotopic Network

Experimentation with the different network architectures clearly demonstrated that a network with frequency division vastly improved the localization performance of the TDNNs (Figure 1). In this case, frequency division was essential to producing a reasonable neural system model that would localize similarly to the human subject across *all* of the different band-pass conditions. For any single band-pass condition, it was found that the TDNN did not require frequency division within its architecture to produce quality solutions when trained *only* on these band-passed sounds.

As mentioned above it was observed that a tonotopic network, one that divides the input frequency channels into different groups and then progressively interconnects the neurons in the higher layers across frequency, was more robust in its localization performance across sounds with variable center-frequency and bandwidth than a simple fully connected network. There are two likely explanations for this observation. One line of reasoning argues that it was easier for the tonotopic network to prevent a narrow band of frequency channels from dominating the localization computation across the entire set of sound stimuli. Or expressed slightly differently, it may have been easier for it to incorporate the relevant information across the different frequency channels. A second line of reasoning argues that the tonotopic network structure (along with the training with variable sounds) encouraged the network to develop meaningful connections for all frequencies.

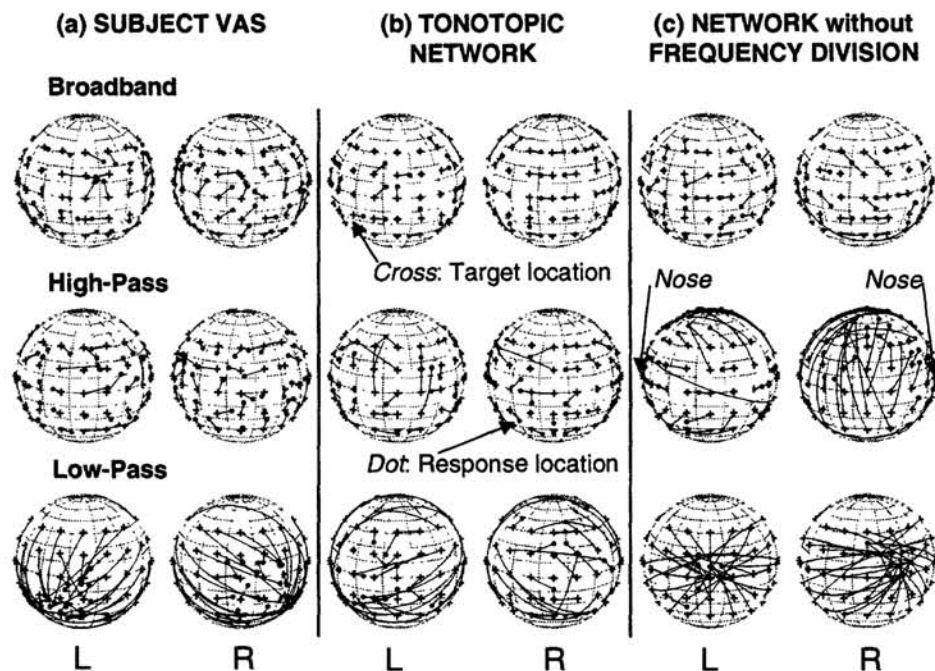

Figure 1: Comparison of the subject's VAS localization performance and the model's localization performance both with and without frequency division. The viewpoint is from an outside observer, with the target location shown by a cross and the response location shown by a black dot.

## 5    Matched Filtering and Sound Localization

A number of previous sound localization models have used a relatively straight-forward matched filter or template matching analysis [9]. In such cases, the ITD and spectrum of a given input sound is commonly cross-correlated with the ITD and spectrum of an entire database of sounds for which the location is known. The location with the highest correlation is then chosen as the optimal source location.

Matched filtering analysis is compared with the localization performance of both the human subject and the neural system model using a bandpass sound with restricted high-frequencies (Figure 2). The matched filtering localizes the sounds much better than the subject or the TDNN model. The matched filtering model used the same number of cochlear channels as the TDNNs and therefore contained the same inherent spectral resolution. This spectral resolution (31 cochlear channels) is certainly less than the spectral resolution of the human cochlea. This shows that although there was sufficient information to localize the sounds from the point of view of matched filtering, neither the human nor TDNN demonstrated such ability in their performance. In order for the TDNN to localize similarly to the matched filtering model, the network weights corresponding to a given location need to assume the form of the filter template for that location. As all of the training sounds were flat-spectrum, the TDNN received no ambiguity as far as the source spectrum was concerned. Thus it is likely that the difference in the distribution of localization responses in Figure 2b, as compared with that in Figure 2c, has been encouraged by using training sounds with random center-frequency and bandwidth, providing a partial explanation as to why the human localization performance is not optimal from a matched filtering standpoint.

**Band-pass (7.6 - 13.3 kHz)**

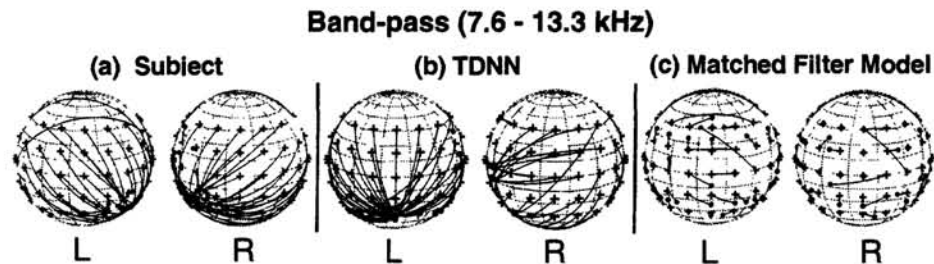

Figure 2: Comparison of the localization performances of the subject, the TDNN model, and a matched filtering model. Details as in Fig. 1.

## 6    Varying Sound Levels and the ILD Cue

The training of the TDNNs was performed in such a fashion, that for any particular location in space, the sound level (67 dB SPL) did not vary by more than 1 dB SPL during repeated presentations of the sound. The localization performance of the neural system model was then examined, using a broadband sound source, across a range of sound levels varying from 60 dB SPL to 80 dB SPL. The spherical correlation coefficient between the target and response locations ([10], values above 0.8 indicate "high" correlation) remained above 0.8 between 60 and 75 dB SPL demonstrating that there was a graceful degradation in localization performance over a range in sound level of 15 dB.

The network was also tested on broadband sounds, 10 dB louder in one ear than the other. The results of these tests are shown in Figure 3 and clearly illustrate that the localization responses were pulled toward the side with the louder sound. While the magnitude of this effect is certainly not human-like, such behaviour suggests that interaural level difference

cues were a prominent and constant feature of the data that conferred a measure of robustness to sound level variations.

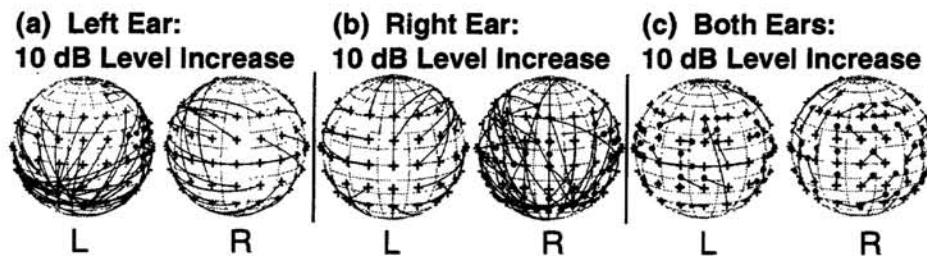

Figure 3: Model's localization performance with a 10 dB increase in sound level: (a,b) monaurally, (c) binaurally.

## 7 Conclusions

A neural system model was developed in which physiological constraints were imposed upon the modeling process: (1) a TDNN model was used to incorporate the important role of spectral-temporal processing in the auditory nervous system, (2) a tonotopic structure was added to the network, (3) the training sounds contained randomly varying center-frequencies and bandwidths. This biologically plausible model provided increased understanding of the role that these constraints play in determining localization performance.

### Acknowledgments

The authors thank Markus Schenkel and André van Schaik for valuable comments. This research was supported by the NHMRC, ARC, and a Dora Lush Scholarship to CJ.

## References

[1] S. Carlile, *Virtual auditory space: Generation and applications.* New York: Chapman and Hall, 1996.

[2] R. H. Gilkey and T. R. Anderson, *Binaural and Spatial Hearing in real and virtual environments.* Mahwah, New Jersey: Lawrence Erlbaum Associates, Publishers, 1997.

[3] C. Jin, M. Schenkel, and S. Carlile, "Neural system identification model of human sound localisation," *(Submitted to J. Acoust. Soc. Am.),* 1999.

[4] S. Hyams and S. Carlile, "After-effects in auditory localization: evidence for channel based processing," *Submitted to the J. Acoust. Soc. Am.,* 2000.

[5] S. Carlile, P. Leong, and S. Hyams, "The nature and distribution of errors in the localization of sounds by humans," *Hearing Research,* vol. 114, pp. 179–196, 1997.

[6] S. Carlile, S. Delaney, and A. Corderoy, "The localization of spectrally restricted sounds by human listeners," *Hearing Research,* vol. 128, pp. 175–189, 1999.

[7] C. Giguère and P. C. Woodland, "A computational model of the auditory periphery for speech and hearing research. i. ascending path," *J. Acoust. Soc. Am.,* vol. 95, pp. 331–342, 1994.

[8] C. Neti, E. Young, and M. Schneider, "Neural network models of sound localization based on directional filtering by the pinna," *J. Acoust. Soc. Am.,* vol. 92, no. 6, pp. 3140–3156, 1992.

[9] J. Middlebrooks, "Narrow-band sound localization related to external ear acoustics," *J. Acoust. Soc. Am.,* vol. 92, no. 5, pp. 2607–2624, 1992.

[10] N. Fisher, I, T. Lewis, and B. J. J. Embleton, *Statistical analysis of spherical data.* Cambridge: Cambridge University Press, 1987.